# Replay, Repair and Consolidation

**Szabolcs Káli**
Institute of Experimental Medicine
Hungarian Academy of Sciences
Budapest 1450, Hungary
kali@koki.hu

**Peter Dayan**
Gatsby Computational Neuroscience Unit
University College London
17 Queen Square, London WC1N 3AR, U.K.
dayan@gatsby.ucl.ac.uk

## Abstract

A standard view of memory consolidation is that episodes are stored temporarily in the hippocampus, and are transferred to the neocortex through replay. Various recent experimental challenges to the idea of transfer, particularly for human memory, are forcing its re-evaluation. However, although there is independent neurophysiological evidence for replay, short of transfer, there are few theoretical ideas for what it might be doing. We suggest and demonstrate two important computational roles associated with neocortical indices.

## 1   Introduction

Particularly since the analysis of subject HM,[1] the suggestion that human memories would *consolidate,*[2] has gripped experimental and theoretical communities. The idea is that storage of some sorts of knowledge (notably declarative information) involves a two-stage process, with memories moving from an initial, temporary, home (usually taken to be the hippocampus), which offers fast acting, but short-lived, plasticity, into a final, permanent resting place (usually the neocortex), whose learning and forgetting are much slower.

Various sources of evidence have been adduced in favor of this proposition. First, it has been suggested that for patients (or animal subjects) who have suffered insults to the hippocampus, recent memories are more compromised than older ones, suggesting that they have yet to be consolidated to cortex.[3,4] Second, the same patients suffer from anterograde amnesia (that is, they cannot lay down new memories), even though many neocortical areas are palpably functioning, and procedural storage (including aversive conditioning and skill learning) works (more) normally.[5] Third, starting with the seminal work of Marr,[6] who (possibly by a mis-calculation[7]) suggested that the hippocampus was just large enough a dynamic RAM as to store one day's events, a variety of theoretical treatments has suggested the possible characteristics and advantages of two-stage procedures.[8–10] This is widely regarded as reaching its apogee in the work of McClelland *et al*,[11] who performed a careful computational analysis of fast and slow learning in connectionist networks. Fourth, and perhaps most compelling, an obvious substrate for replay to cortex is provided by the neurophysiologically observed[12–14] reactivation during slow wave and REM sleep of patterns of (rat) hippocampal neuronal firing observed during times when the subject is awake and behaving, together with evidence of at least some coordination between hippocampal and neocortical states during this reactivation.[15]

The first and third of these evidentiary foundations are currently under active debate, specially for episodic memories (*ie* autobiographical memories for happenings). Solid evidence that hippocampal damage really spares memories for distant events compared with

those for recent ones is extremely sparse, and the relevance of infra-human studies is put into question by the orders-of-magnitude differences in the memory time-scales shown between humans and animals.[16] The modeling studies are also more ambiguous than they might seem, since their most convincing focus is on the tribulations of catastrophic interference.[17] That is, slow learning is necessary in systems with rich distributed or population coding because changes in synaptic efficacies occasioned by incorporating new information can easily overwrite the neural substrate for the storage of old information (the hoary stability-plasticity dilemma[18]). This catastrophic interference can be avoided by re-storing old patterns (or something equivalent[10,19]) at the same time as storing new information. Thus, according to these schemes, patterns are stored wholesale in the hippocampus when they first appear, and are continually read back to cortex to cause plasticity along with the new information. However, if the hippocampus is permanently required to prevent a catastrophe, then, first, there is no true consolidation: if neocortical plasticity is not inhibited by hippocampal damage,[20] then its integrity is permanently required to prevent degradation; and, second, what is the point of consolidation – couldn't the hippocampus suffice by itself? This is particularly compelling in the case of episodes, since they are intrinsically isolated events. We came to a realization of this through development of our own model for consolidation,[21] whose behavior convinced us of a flaw in our thinking. This second point lies exactly at the heart of the perspective espoused by Nadel and Moscovitch,[16] amongst others. They regard the hippocampus as the final point of storage for all episodic memory, and permanently required for its recall. Of course, this idea equally well accounts for the second strand of evidence above about anterograde amnesia.

If the hippocampus stores patterns permanently, what could the point be of replay? Here, we consider two roles, both associated with concerns about the pattern matching process at the heart of retrieval from the hippocampus. One is a new take on catastrophic interference, arguing that replay is necessary to keep the patterns stored in the hippocampus in register with the evolving cortical representation, so that they can still be recalled (and interpreted) correctly even though the cortical code may have changed since they were stored. The other computational role for replay is a new take on indexing, arguing that the cortical patterns that should lead to retrieval of a hippocampal memory are not only close syntactic relatives of the pattern that was originally stored, *ie* patterns whose actual neural code is similar, but also patterns that are close semantic relatives, *ie* patterns that are closely related via the network of semantic relationships that is stored in neocortex. In this scheme, the role of replay is building an index to the memory, effectively a form of recognition model.[22]

We first discuss briefly our existing model of consolidation,[21] and its failings. Section 3 treats the repair of hippocampal indexing in the light of the vicissitudes of semantic change. Section 4 sketches our account of the semantic elaboration of the index.

## 2   Semantic and Episodic Memory

Figure 1 shows our existing account of the interaction between the neocortex and the hippocampus in semantic and episodic memory.[21] The neocortex is separated into 'lower' areas $(\mathbf{x}^A, \mathbf{x}^B, \ldots)$ which are connected via bi-directional, variable, weights $\mathbf{W}$ with an entorhinal/parahippocampal (EP) area $(\mathbf{y})$, and collectively act as a restricted Boltzmann machine (RBM), trained in an unsupervised manner, using contrastive divergence.[23] It learns a model of the statistical relationships amongst the inputs, so that it can produce samples from conditional probability distributions such as $P[\mathbf{x}^C | \mathbf{x}^A, \mathbf{x}^B; \mathbf{W}]$. The conventional interpretation for this is as a model of semantic memory – the generic facts of the world, stripped of information about the time and place and other circumstances under which they were learnt. However, the individual patterns on which the semantic learning is based are treated as episodic patterns, which should be recalled wholesale. One main contribution of that work was to put episodic and semantic information into such particular correspondence.

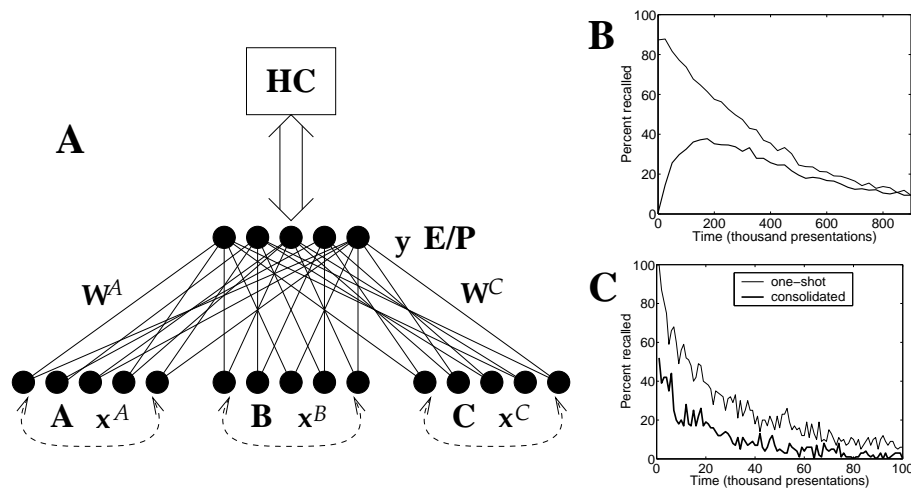

**Figure 1**: **(A)** Model architecture. All units in neocortical areas A, B, and C are connected to all units in area E/P through bidirectional, symmetric weights, but connections between units in the input layer are restricted to the same cortical area. Each neocortical area contains 100 binary units. The hippocampus (HC) is not directly implemented, but it can influence and store the patterns in EP. All communication between the HC and the input areas is via area EP. **(B)** The consolidation of episodic memories. Recall performance on specific (episodic) patterns as a function of time between the initial presentation of the episodic pattern and testing (or, equivalently, time between training and lesion in hippocampals) in the simulations. **(C)** Extinction of an episode due to semantic training, in the isolated neocortical network trained to asymptotic performance on the episodic pattern (thin line), and directly after the removal of the hippocampus from the full network, for a pattern which has been hippocampally "consolidated" for 250,000 presentations (thick line).

In this previous model, the hippocampus acts as a fast-learning repository for the EP representation of patterns that have been (relatively recently) experienced, and plays two roles: aiding recall and training the neocortex. The hippocampus improves recall by performing pattern completion on the EP representations induced by partial or noisy inputs $\mathbf{x}$, thus finding the nearest matching stored $\mathbf{y}$. In turn, this, through neocortical semantic knowledge, engenders recall of an appropriate $\mathbf{x}$. The hippocampus trains the neocortex in an off-line (sleep) mode, reporting the patterns that it has stored to the neocortex to give the latter's incremental plasticity the opportunity to absorb the new information. Given hippocampal damage, patterns that have been repeatedly replayed to cortex by the hippocampus (*ie* older patterns) have a greater chance of being recalled correctly through neocortical inference than patterns that were learned more recently, and are therefore still dependent for their recall on the integrity of the hippocampus.

Figure 1B shows the basic consolidation phenomenon in this model. The upper (thin) curve shows how well on average the full model can recall whole items from a partial cue as a function of time since the item was stored; the lower (thick) curve shows the same in the case that the hippocampal contribution is eliminated immediately before testing. This is the standard inverted U-shaped curve of graded retrograde amnesia, with distant memories spared compared with recent ones. However, figure 1C reveals what is really going on. Both curves show how the neocortical network forgets particular episodic patterns as a function of continued semantic training. Thick/thin lines are with/without prior consolidation using the hippocampus. Consolidation clearly does *not* help the longevity of the memory – if anything, it actually impedes it. This is essentially because the cortical code changes slowly over presentations. Thus, first, the hippocampus is mandatorily required if memories are to be preserved – the forgetting curve for the normals in figure 1B is actually

dominated by hippocampal forgetting. Second, the inverted U-shaped curve in figure 1B arises because testing happens immediately after hippocampal removal. The same curves plotted for successive times after removal would show catastrophic memory failure.

Memories might turn out to be stabilized in the face of hippocampal damage in other ways.[21] For instance, cortical plasticity might be suppressed, if the hippocampus reports unfamiliarity as a plasticizing signal. This is somewhat unlikely, since various forms of continued plasticity remain active.[3,20] Alternatively, there might be synaptic stabilizing mechanisms in the cortex such that synapses come never to change. This is certainly possible, but does not explain how recall can survive changes in the cortical code.

In sum, the model turns out to illustrate the key problem with standard theory of memory transfer for episodes. We are thus forced to start from the possibility that the hippocampus might indeed be a permanent repository, and reconsider the issue of replay and consolidation in the resulting light. In this new scheme, there is still a critical role for replay, but one that is focused on the *indexing* relationship between neocortical and hippocampal representations rather than on writing into cortex the contents of the hippocampus.

## 3   Maintaining Access to Episodes

Consider the fate of an episode that is stored in the hippocampus. In a hierarchical network where the hippocampus is directly connected only to the topmost areas, successful recall of such an episode depends on the correspondence between low- and high-level cortical areas embodied by the neocortical network. This dependence actually has two related components. First, the high-level neocortical representation of the recall cue needs to be effective in activating the correct hippocampal memory trace; second, the high-level representation activated by hippocampal recall should effect the recall of the appropriate components of the corresponding episode in lower level areas as well. These are both aspects of indexing.

The neocortical network is the substrate of neocortical learning, reflecting, for instance, refinement of the existing semantic representation, changes in input statistics, or acquisition of a new semantic domain. Such plasticity may disrupt the recall of stored episodic patterns by changing the correspondence between the input areas and EP. Thus, if the brain is still to be able to recall hippocampally stored episodes, it either needs to maintain the correspondence between the low-level and EP representations of the episodes by restricting neocortical learning (achieved in the previous model by having the hippocampus replay its old episodic patterns along with the new semantic patterns governing continued neocortical plasticity), or it needs to update the connections between the hippocampus and EP such that the hippocampally stored pattern continues to match the EP representation of the input pattern corresponding to the episode. The first of these possibilities may restrict the learning abilities of the neocortical network. However, replay can be used to allow the connections into and out of the hippocampus to track the changing neocortical representational code.

In order to assess the effect of neocortical learning on the recall of previously stored episodes, either in the presence or absence of replay, the following paradigm was employed. We started training the neocortical network by presenting to the input areas random combinations of valid patterns (20 independently generated random binary patterns for each area). After a moderate amount of such general training (10,000 pattern presentations total), the EP representations of 8 particular input patterns were associated with corresponding stored hippocampal traces, forming a set of stored episodes. The quality of recall for these episodes was then monitored while general training continued. Figure 2A shows as a function of the length of general semantic training the percentage of correct recall for the episodes stored after 10,000 presentations. The main plot is an average over all 8 episodes; the smaller plots show some individual episodes. Clearly, neocortical learning comes to erase the route to recall, even though the episode remains perfectly stored in the hippocampus throughout.

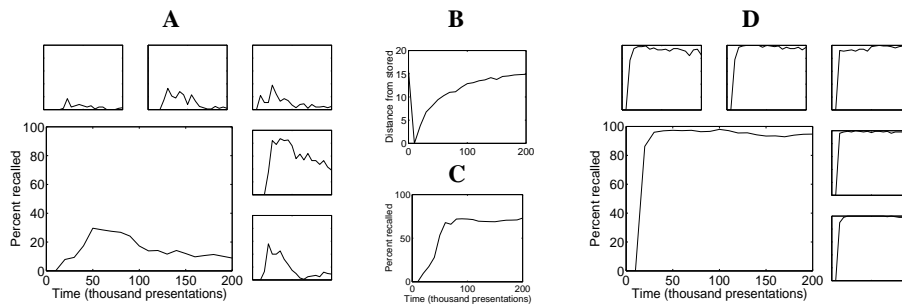

**Figure 2**: How semantic training affects episodic recall for patterns stored after the first 10,000 presentations **(A)** without replay and **(D)** with the correspondence between hippocampal and neo-cortical representations updated during off-line replay. The larger graphs are averages over all stored episodes, while the smaller graphs are for individual episodes. Recall was assessed by presenting partial episodic patterns (the original activations replaced by random patterns in one of the input areas), performing hippocampal pattern completion in EP if the distance from a stored EP representation was less than 20, and then performing 20 full iterations of Gibbs sampling in the neocortical network with the cue areas clamped. A resulting distance of less than 5 from the target pattern was considered a match. **(B)** and **(C)** analyze the reasons why episodic recall breaks down in **(A)**. **(B)** shows how the EP representation of stored episodes drifts away from the original stored patterns. **(C)** shows how well recall works if it starts from the stored EP representation of the episode.

Figure 2B,C indicate the reasons for this behavior. Figure 2B shows that semantic learning after the storage of the episode causes the EP representation of the episode to move away from the version with which the stored hippocampal trace is associated. The magnitude of this change is such that, eventually, even the full original episode may fail to activate the corresponding hippocampal memory trace. The effect of representational change on hippocampally directed recall in the input areas is milder in our case, as seen in Figure 2C; provided that the correct hippocampal trace does get activated, the full episode can be successfully recalled most of the time. However, this component accounts for the relatively slower initial rise of episodic recall in Figure 2A (compare with Figure 2D), as well as some of the variability between patterns in Figure 2A (data not shown).

In the "replay" condition, the general training was interleaved with epochs of hippocampally initiated replay, assumed to take place during sleep. Within these epochs, the memory traces stored in the hippocampus get activated at random, which leads to the reactivation of the associated EP pattern, which in turn reactivates the input areas according to the existing semantic mapping. The resulting pattern may be different from the one that initially gave rise to the stored episode, due to subsequent changes in the neocortical connections. However, assuming that the neocortical semantic representation has not changed fundamentally since the last time that particular episode was replayed (or when it was established), the input representation resulting from replay should be close to the current low level representation of that particular episode. Indeed, maintaining this representational proximity exactly sets the requirement for the frequency of replay of the episodes.

As in our previous model, we assume that the local connections within each neocortical area implement a local attractor structure, which, in the absence of feedforward activation, restricts activity patterns within that area to those that correspond to valid input patterns. These local attractors turn feedback activation which is close to a valid pattern (namely, the original episode) into an exact version of that pattern. Such an off-line reconstruction of the low-level representation of stored episodes may then support a wide variety of memory processes (including the previous model's focus on gradually incorporating the information carried by that episode into the neocortical knowledge base[11,21]). Here we focus on its use for maintenance of the episodic index. To this end, starting from the reconstructed

episode, the semantic correspondence between the different levels is employed in the feed-forward direction in order to determine the up-to-date EP representation of the episode. This EP pattern is then associated with the stored hippocampal episode which initiated the replay, so that the hippocampal and input level representations of the episode are again in register. Figure 2B demonstrates the efficacy of replay: the hippocampally stored episode now remains tied to the (shifting) EP representation of the episode, and episodic replay stays at high levels despite substantial changes in the neocortical network.

## 4 Index Extension

Another important potential role for replay is extending the semantic aspects of the in-dexing scheme. It should be possible to retrieve episodic memories on the basis of all input patterns to which they are closely related through the network of cortical semantic knowledge. At present, this can happen only if the cortex produces similar EP *codes* for all those input patterns that are semantically related. However, requiring that all semantic proximity be coded by syntactic proximity in essentially one single layer, is far too strin-gent a requirement. Rather, we should expect that the bulk of semantic information lives in synapses that are invisible to this layer, *ie* connections within and between lower layers, and this information must also influence indexing.

One way to extend semantic indexing involves on-line sampling. That is, probabilistic updating in the cortical semantic network starting from a given input pattern is the canonical way of exploring the semantic neighborhood of an input. One can imagine doing this in a on-line manner, spurred by an input. Over sampling, the cortical pattern and its EP code change together, providing the opportunity for a match to be made between the EP activity and the contents of episodic memory. These sampling dynamics would allow the recall of semantically relevant episodes, even if their explicit code is rather distant.

The role for replay in this process is to allow the semantic index to be extended through off-line rather than on-line sampling starting from the episodic patterns stored in the hip-pocampus. It is thus analogous to Sutton's[24] use of replay in his DYNA architecture, in which an internal model of a Markov decision process is used to erase inconsistencies in a learned value function, and also to the wake-sleep algorithm's[22] use of sleep sam-pling to learn a recognition model. For the latter, off-line sampling ensures that inputs can be mapped using a feedforward network, into codes associated with a generative model, rather than relying on sluggish statistical or dynamical methods for inverting the generative model, such as Gibbs sampling or its mean-field approximations. The main requirement is for a further plastic layer between EP and CA3 (presumably the perforant path) so that when replay based on an episode leads to a semantically, but not syntactically, related pat-tern, then the EP code for that pattern can induce hippocampal recall of the episode.

Figure 3 illustrates this use of replay in a highly simplified case (subject to the limita-tions of the RBM). Here, there are 3 modules of 16 units, each with 4 possible patterns, and a semantic structure such that $P[\mathbf{x}_i^B|\mathbf{x}_i^A] = 0.75$; $P[\mathbf{x}_{i+1}^B|\mathbf{x}_i^A] = 0.15$; $P[\mathbf{x}_{i+2}^B|\mathbf{x}_i^A] = P[\mathbf{x}_{i+3}^B|\mathbf{x}_i^A] = 0.05$ (with wrap-around, so, *eg,* $\mathbf{x}_5^B \equiv \mathbf{x}_1^B$) and $\mathbf{x}^C$ independent of the choice in $\mathbf{x}^A$ and $\mathbf{x}^B$. Figure 3A shows the covariance matrix of the activities of the 100 EP units to the 64 possible input patterns (arranged lexicographically). The relatedness of the EP representation of related patterns is clear in the rich structure of this matrix – this shows the extent of the explicit code learnt by the RBM. However, this code does not make indexing perfect. Imagine that $E_1 = \{\mathbf{x}_1^A, \mathbf{x}_2^B, \mathbf{x}_1^C\}$ and $E_2 = \{\mathbf{x}_3^A, \mathbf{x}_4^B, \mathbf{x}_4^C\}$ have been stored as episodic patterns. That is, their EP representations are stored in the hippocampus and are available for recall and replay. We may expect to retrieve $E_1$ from its semantic relation $E_* = \{\mathbf{x}_1^A, \mathbf{x}_1^B, \mathbf{x}_1^C\}$. Figure 3B shows the explicit proximity (inverse square distance, see caption) of the EP representations of the 64 input patterns to the EP representation of $E_1$. Although $E_*$ is close, so are many other patterns that are not nearly so closely semantically related. For instance, $\{\mathbf{x}_1^A, \mathbf{x}_3^B, \mathbf{x}_1^C\} \equiv 131$ and $\{\mathbf{x}_1^A, \mathbf{x}_4^B, \mathbf{x}_1^C\} \equiv 141$ are closer.

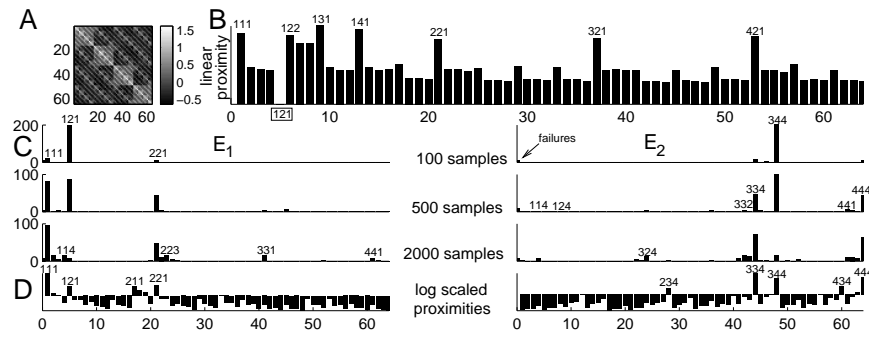

**Figure 3**: Index expansion. Plots relate to the 3-module network. Conventions: 1–64 denote the 64 possible input patterns or their EP representations. $111 \equiv \{\mathbf{x}_1^A, \mathbf{x}_1^B, \mathbf{x}_1^C\}$, $112 \equiv \{\mathbf{x}_1^A, \mathbf{x}_1^B, \mathbf{x}_2^C\}$, *etc.* In (C), the entry for 0 shows patterns that are not within Hamming distance of 1 of any input pattern. For this simulation, for reasons of simulation time, the input patterns were chosen to be orthogonal; the hidden unit representations were nevertheless highly non-orthogonal; 5 iterations of Gibbs sampling were used during RBM learning. The weights associated with the network are not over-trained. A) The covariance matrix of the EP representation of the 64 possible patterns. The banding shows the semantic structure (see text), but, as seen in (B), only weakly. B) The proximities $(1/\|\mathbf{y}_* - \mathbf{y}_{E_1}\|^2)$ of the EP representations ($\mathbf{y}_*$) for all the patterns to that for $E_1$ (the entry for $\mathbf{y}_{E_1}$ is blank; see boxed 121). The numbers refer to the patterns as in the convention described. Despite the covariance structure in (A), the syntactic representation of semantic closeness is weak: 121 is not closely related to 421, for instance. Thus, episodic recall would be imperfect. Ratio of max-min proximity (bar 121) is 4. C) Three stages of (unclamped) Gibbs sampling starting (250 times each) from the hippocampally replayed EP representations of $E_1$ (left column) and $E_2$ (right column). Here, we determine to which (if any — thus the 'failure' entry 0) of the 64 possible input patterns, the sampled activities of the visible units are closest, and plot histograms of the resulting frequencies. After only few iterations, 121 and 344 still dominate; after more, the semantically close patterns 111 and 221 dominate for $E_1$ and 334 and 444 for $E_2$. D) Logarithmically scaled proximities following delta-rule learning for the mapping from EP representations of the patterns in (C) to $E_1$ and $E_2$ respectively. Now, the remapped EP representations of semantically relevant inputs are vastly closer to their associated episodic memories. Ratios of max-min proximities are 14000 ($E_1$) and 7000 ($E_2$).

Figure 3C shows the course of replay. The two columns show histograms of the patterns retrieved in the visible layer after $100, 500, 2000$ rounds of Gibbs sampling starting (250 times) from the hippocampal representation of $E_1$ (left) and $E_2$ (right). The network has learnt much about the semantic relationships, although it is far from perfect (over-training seems to make it *worse,* for reasons we do not understand), and equally likely patterns are not generated exactly equally often.[21] The 0 columns of these histograms show how many sampled visible patterns are not close to one of the 64 valid inputs; this happens only rarely. During replay, the EP representation of these semantically-related patterns is then available so that a model mapping EP to an appropriate input to the hippocampal pattern matching process can be learnt. Figure 3D shows how this affects the proximities for a model trained using the delta rule. Again, left and right columns are for $E_1$ and $E_2$; now the semantic associates of these patterns are mapped into inputs to the hippocampal pattern matching process that are far nearer (note the logarithmic scale) to the stored representations of $E_1$ and $E_2$, and so the episodes can be appropriately retrieved from their semantic cousins.

## 5   Discussion

The important, but narrow, issue of whether episodic memories can ever be recalled without the hippocampus has polarized theoretical ratiocination about memory replay, a phe-

nomenon for which there is increasing neurophysiological evidence. This polarization has hindered the field from studying the wider computational context of replay. In this paper, we have considered two particular aspects of the consolidation of the *indexing* relationship between semantic memory (in the neocortex) and episodic memory (in the hippocampus). We showed how replay could be used to maintain the index in the face of on-going neocortical plasticity, and to broaden it in the light of neocortical semantic knowledge that is not directly accessible through the explicit code in the upper layers of cortex. Unlike memory consolidation, neither of these involves neocortical plasticity during replay. There may yet be many other computations that can be accomplished through replay.

Broadening the index poses an interesting, only incompletely answered, theoretical question about the metrics of memory. The semantic model can be seen as a sort of manifold in the space of all inputs; the episodes as particular points on the manifold; and retrieval as finding the closest episodes to a presented cue, according to a distance function that involves mapping the cue to the manifold, and mapping between points on the manifold. Despite some theoretical suggestions,[25] it is not clear how the semantic model specifies these distances. Our pragmatic solution was to replay the episodes and rely on the transience of the Markov chain induced by Gibbs sampling to produce semantic cousins with which it should be related. It would be desirable to consider more systematic approaches.

Our model involves interaction between a hippocampal store for episodes and a neocortical store for semantics. However, the computational issues about indexing apply with the same force if the episodes are actually stored separately elsewhere, such as in more frontal structures (McClelland, personal communication). There are equal opportunities for these areas to induce replay, and thus improve the index. What now seems unlikely, despite our best earlier efforts, is that the problems of indexing can be circumvented by storing the episodes wholly within the semantic network. By itself, this solves nothing.

### Acknowledgements

We are very grateful to Jay McClelland for helpful discussions. Funding was from the Hungarian Academy of Sciences and the Gatsby Charitable Foundation.

### References

[1]  W. Scoville and B. Milner, J Neurol Neurosurg Psychiatry **20**, 11 (1957).

[2]  T. Ribot, *Les maladies de la memoire*, Appleton-Century-Crofts, New York, 1881.

[3]  L. R. Squire, Psychol Rev **99**, 195 (1992).

[4]  L. R. Squire, R. E. Clark, and B. J. Knowlton, Hippocampus **11**, 50 (2001).

[5]  A. R. Mayes and J. J. Downes, Memory **5**, 3 (1997).

[6]  D. Marr, Philos Trans R Soc Lond B Biol Sci **262**, 23 (1971).

[7]  D. J. Willshaw and J. T. Buckingham, Philos Trans R Soc Lond B Biol Sci **329**, 205 (1990).

[8]  P. Alvarez and L. R. Squire, Proc Natl Acad Sci U S A **91**, 7041 (1994).

[9]  J. M. Murre, Memory **5**, 213 (1997).

[10]  R. M. French, Connection Science **9**, 353 (1997).

[11]  J. L. McClelland, B. L. McNaughton, and R. C. O'Reilly, Psychol Rev **102**, 419 (1995).

[12]  M. A. Wilson and B. L. McNaughton, Science **265**, 676 (1994).

[13]  W. E. Skaggs and B. L. McNaughton, Science **271**, 1870 (1996).

[14]  K. Louie and M. A. Wilson, Neuron **29**, 145 (2001).

[15]  A. G. Siapas and M. A. Wilson, Neuron **21**, 1123 (1998).

[16]  L. Nadel and M. Moscovitch, Curr Opin Neurobiol **7**, 217 (1997).

[17]  M. McCloskey and N. J. Cohen, in *The psychology of learning and motivation, vol 24*, edited by G. Bower, 109–165, Academic Press, New York, 1989.

[18]  G. A. Carpenter and S. Grossberg, Trends Neurosci **16**, 131 (1993).

[19]  A. Robins, Connection Science **8**, 259 (1996).

[20]  F. Vargha-Khadem et al., Science **277**, 376 (1997).

[21]  S. Káli and P. Dayan, in *NIPS 13*, edited by T. K. Leen, T. G. Dietterich, and V. Tresp, 24–30, MIT Press, Cambridge, 2001.

[22]  G. E. Hinton, P. Dayan, B. J. Frey, and R. M. Neal, Science **268**, 1158 (1995).

[23]  G. E. Hinton, Neural Computation, **14** (2002).

[24]  R. S. Sutton, in *Machine Learning: Proceedings of the Seventh International Conference*, 216–224, 1990.

[25]  L. K. Saul, in *NIPS 9*, edited by M. C. Mozer, M. I. Jordan, and T. Petsche, 267–273, MIT Press, London, UK, 1997.
